# Maximum Likelihood and the Information Bottleneck

**Noam Slonim    Yair Weiss**
School of Computer Science & Engineering,
Hebrew University, Jerusalem 91904, Israel
{noamm,yweiss}@cs.huji.ac.il

## Abstract

The information bottleneck (IB) method is an information-theoretic formulation for clustering problems. Given a joint distribution $p(x, y)$, this method constructs a new variable $T$ that defines partitions over the values of $X$ that are informative about $Y$. Maximum likelihood (ML) of mixture models is a standard statistical approach to clustering problems. In this paper, we ask: how are the two methods related ? We define a simple mapping between the IB problem and the ML problem for the multinomial mixture model. We show that under this mapping the problems are strongly related. In fact, for uniform input distribution over $X$ or for large sample size, the problems are mathematically equivalent. Specifically, in these cases, every fixed point of the IB-functional defines a fixed point of the (log) likelihood and vice versa. Moreover, the values of the functionals at the fixed points are equal under simple transformations. As a result, in these cases, every algorithm that solves one of the problems, induces a solution for the other.

## 1   Introduction

Unsupervised clustering is a central paradigm in data analysis. Given a set of objects $X$, one would like to find a partition $T(X)$ which optimizes some score function. Tishby *et al.* [1] proposed a principled information-theoretic approach to this problem. In this approach, given the joint distribution $p(x, y)$, one looks for a compact representation of $X$, which preserves as much information as possible about $Y$ (see [2] for a detailed discussion).

The mutual information, $I(X; Y)$, between the random variables $X$ and $Y$ is given by [3] $I(X; Y) = \sum_{x \in X, y \in Y} p(x)p(y|x) \log \frac{p(y|x)}{p(y)}$ . In [1] it is argued that both the compactness of the representation and the preserved relevant information are naturally measured by mutual information, hence the above principle can be formulated as a trade-off between these quantities. Specifically, Tishby *et al.* [1] suggested to introduce a compressed representation $T$ of $X$, by defining $q(t|x)$. The compactness of the representation is then determined by $I(T; X)$, while the quality of the clusters, $T$, is measured by the fraction of information they capture about $Y$, $I(T; Y)/I(X; Y)$. The IB problem can be stated as finding a (stochastic) mapping $q(t|x)$ such that the IB-functional $\mathcal{L} = I(T; X) - \beta I(T; Y)$ is minimized, where $\beta$ is a positive Lagrange multiplier that determines the trade-off between compression and precision. It was shown in [1] that this problem has an exact optimal (formal) solution without any assumption about the origin of the joint distribution $p(x, y)$.

The standard statistical approach to clustering is mixture modeling. We assume the mea-

surements $y$ for each $x$ come from one of $|T|$ possible statistical sources, each with its own parameters $\Theta_t$ (e.g. $\mu_t, \sigma_t$ in Gaussian mixtures). Clustering corresponds to first finding the maximum likelihood estimates of $\Theta_t$ and then using these parameters to calculate the posterior probability that the measurements at $x$ were generated by each source. These posterior probabilities define a "soft" clustering of $X$ values.

While both approaches try to solve the same problem the viewpoints are quite different. In the information-theoretic approach no assumption is made regarding how the data was generated but we assume that the joint distribution $p(x, y)$ is known exactly. In the maximum-likelihood approach we assume a specific generative model for the data and assume we have samples $n(x, y)$, not the true probability.

In spite of these conceptual differences we show that under a proper choice of the generative model, these two problems are strongly related. Specifically, we use the multinomial mixture model (a.k.a the one-sided [4] or the asymmetric clustering model [5]), and provide a simple "mapping" between the concepts of one problem to those of the other. Using this mapping we show that in general, searching for a solution of one problem induces a search in the solution space of the other. Furthermore, for uniform input distribution $p(x)$ or for large sample sizes, we show that the problems are mathematically equivalent. Hence, in these cases, any algorithm which solves one problem, induces a solution for the other.

## 2   Short review of the IB method

In the IB framework, one is given as input a joint distribution $p(x, y)$. Given this distribution, a compressed representation $T$ of $X$ is introduced through the stochastic mapping $q(t|x)$. The goal is to find $q(t|x)$ such that the IB-functional, $\mathcal{L} = I(T; X) - \beta I(T; Y)$ is minimized for a given value of $\beta$.

The joint distribution over $X, Y$ and $T$ is defined through the IB Markovian independence relation, $T \leftarrow X \rightarrow Y$. Specifically, every choice of $q(t|x)$ defines a specific joint probability $q_{IB}(x, y, t) = p(x, y)q(t|x)$. Therefore, the distributions $q(t)$ and $q(y|t)$ that are involved in calculating the IB-functional are given by

$$\begin{cases} q(t) = \sum_{x,y} q_{IB}(x, y, t) = \sum_x p(x)q(t|x) \\ q(y|t) = \frac{1}{q(t)} \sum_x q_{IB}(x, y, t) = \frac{1}{q(t)} \sum_x p(x, y)q(t|x) \ . \end{cases} \tag{1}$$

In principle every choice of $q(t|x)$ is possible but as shown in [1], if $q(t)$ and $q(y|t)$ are given, the choice that minimizes $\mathcal{L}$ is defined through,

$$q(t|x) = \frac{q(t)}{Z(\beta, x)} e^{-\beta D_{KL}(p(y|x)\|q(y|t))} \ , \tag{2}$$

where $Z(\beta, x)$ is the normalization (partition) function and $D_{KL}(p\|q) = \sum p \log \frac{p}{q}$ is the Kullback-Leibler divergence. Iterating over this equation and the *IB-step* defined in Eq.(1) defines an iterative algorithm that is guaranteed to converge to a (local) fixed point of $\mathcal{L}$ [1].

## 3   Short review of ML for mixture models

In a multinomial mixture model, we assume that $Y$ takes on discrete values and sample it from a multinomial distribution $\theta(y|t(x))$, where $t(x)$ denotes $x$'s label. In the one-sided clustering model [4] [5] we further assume that there can be multiple observations $y$ corresponding to a single $x$ but they are all sampled from the same multinomial distribution. This model can be described through the following generative process:

- For each $x$ choose a unique label $t(x)$ by sampling from $\pi(t)$.
- For $l = 1 : N$
  - choose $x_l$ by sampling from $\gamma(x)$.
  - choose $y_l$ by sampling from $\theta(y|t(x_l))$ and increase $n(x_l, y_l)$ by one.

Let $\vec{t} = (t_1, ..., t_{|X|})$ denotes the random vector that defines the (typically hidden) labels, or topics for all $x \in X$. The complete likelihood is given by:

$$
\begin{aligned}
p(x, y, \vec{t} : \pi, \theta, \gamma) &= \Pi_{i=1}^{|X|} \pi(t(x_i)) \Pi_{l=1}^{N} \gamma(x_l) \theta(y_l | t(x_l)) \qquad (3) \\
&= \Pi_{i=1}^{|X|} \pi(t(x_i)) \Pi_{i=1}^{|X|} \Pi_{j=1}^{|Y|} [\gamma(x_i) \theta(y_j | t(x_i))]^{n(x_i, y_j)} , \qquad (4)
\end{aligned}
$$

where $n(x_i, y_j)$ is a count matrix. The (true) likelihood is defined through summing over all the possible choices of $\vec{t}$,

$$
L(n(x, y) : \pi, \theta, \gamma) = \sum_{\vec{t}} p(x, y, \vec{t} : \pi, \theta, \gamma) . \qquad (5)
$$

Given $n(x, y)$, the goal of ML estimation is to find an assignment for the parameters $\pi(t), \theta(y|t)$ and $\gamma(x)$ such that the likelihood is (at least locally) maximized. Since it is easy to show that the ML estimate for $\gamma(x)$ is just the empirical counts $n(x)/N$ (where $n(x) = \sum_y n(x, y)$), we further focus only on estimating $\pi, \theta$.

A standard algorithm for this purpose is the EM algorithm [6]. Informally, in the $E$-step we replace the missing value of $t(x)$ by its distribution $p(t(x)|y(x))$ which we denote by $q_x(t)$. In the $M$-step we use that distribution to reestimate $\pi, \theta$. Using standard derivation it is easy to verify that in our context the $E$-step is defined through

$$
\begin{aligned}
q_x(t) &= k(x) \pi(t) e^{n(x) \sum_y n(y|x) \log \theta(y|t)} \qquad (6) \\
&= k_2(x) \pi(t) e^{n(x)[\sum_y n(y|x) \log \theta(y|t) - \sum_y n(y|x) \log n(y|x)]} \qquad (7) \\
&= k_2(x) \pi(t) e^{-n(x) D_{KL}(n(y|x) \| \theta(y|t))} , \qquad (8)
\end{aligned}
$$

where $k(x)$ and $k_2(x)$ are normalization factors and $n(y|x) = \frac{n(x,y)}{n(x)}$. The $M$-step is simply given by

$$
\begin{cases}
\pi(t) \propto \sum_x q_x(t) \\
\theta(y|t) \propto \sum_x n(x, y) q_x(t) .
\end{cases} \qquad (9)
$$

Iterating over these EM steps is guaranteed to converge to a local fixed point of the likelihood. Moreover, every fixed point of the likelihood defines a fixed point of this algorithm.

An alternative derivation [7] is to define the free energy functional:

$$
\begin{aligned}
F(n(x, y) : q, \pi, \theta) &= -\sum_{t,x} q_x(t) \left[ \log \pi(t) + \sum_y n(x, y) \log \theta(y|t) \right] \qquad (10) \\
&+ \sum_{t,x} q_x(t) \log q_x(t) . \qquad (11)
\end{aligned}
$$

The $E$-step then involves minimizing $F$ with respect to $q$ while the $M$-step minimizes it with respect to $\pi, \theta$. Since this functional is bounded (under mild conditions), the EM algorithm will converge to a local fixed point of $F$ which corresponds to a fixed point of the likelihood. At these fixed points, $F$ will become identical to $-\log L(n(x, y) : \pi, \theta)$.

## 4 The ML ↔ IB mapping

As already mentioned, the IB problem and the ML problem stem from different motivations and involve different "settings". Hence, it is not entirely clear what is the purpose of "mapping" between these problems. Here, we define this mapping to achieve two goals. The first is theoretically motivated: using the mapping we show some mathematical equivalence between both problems. The second is practically motivated, where we show that algorithms designed for one problem are (in some cases) suitable for solving the other.

A natural mapping would be to identify each distribution with its corresponding one. However, this direct mapping is problematic. Assume that we are mapping from ML to IB. If we directly map $q_x(t), \pi(t), \theta(y|t)$ to $q(t|x), q(t), q(y|t)$, respectively, obviously there is no guarantee that the IB Markovian independence relation will hold once we complete the mapping. Specifically, using this relation to extract $q(t)$ through Eq.(1) will in general result with a different prior over $T$ then by simply defining $q(t) = \pi(t)$. However, we notice that once we defined $q(t|x)$ and $p(x, y)$, the other distributions could be extracted by performing the IB-step defined in Eq.(1). Moreover, as already shown in [1], performing this step can only improve (decrease) the corresponding IB-functional. A similar phenomenon is present once we map from IB to ML. Although in principle there are no "consistency" problems by mapping directly, we know that once we defined $q_x(t)$ and $n(x, y)$, we can extract $\pi$ and $\theta$ by a simple $M$-step. This step, by definition, will only improve the likelihood, which is our goal in this setting. The only remaining issue is to define a corresponding component in the ML setting for the trade-off parameter $\beta$. As we will show in the next section, the natural choice for this purpose is the sample size, $N = \sum_{x,y} n(x, y)$.

Therefore, to summarize, we define the $ML \leftrightarrow IB$ mapping by

$$q_x(t) \leftrightarrow q(t|x), \quad \frac{1}{N}n(x, y) \leftrightarrow p(x, y), \quad N \leftrightarrow r\beta , \tag{12}$$

where $r$ is a positive (scaling) constant and the mapping is completed by performing an IB-step or an $M$-step according to the mapping direction. Notice that under this mapping, every search in the solution space of the IB problem induces a search in the solution space of the ML problem, and vice versa (see Figure 2).

**Observation 4.1** *When $X$ is uniformly distributed (i.e., $n(x)$ or $p(x)$ are constant), the $ML \leftrightarrow IB$ mapping is equivalent for a direct mapping of each distribution to its corresponding one.*

This observation is a direct result from the fact that if $X$ is uniformly distributed, then the IB-step defined in Eq.(1) and the $M$-step defined in Eq.(9) are mathematically equivalent.

**Observation 4.2** *When $X$ is uniformly distributed, the EM algorithm is equivalent to the IB iterative optimization algorithm under the $ML \leftrightarrow IB$ mapping with $r = |X|$.*

Again, this observation is a direct result from the equivalence of the IB-step and the $M$-step for uniform prior over $X$. Additionally, we notice that in this case $n(x) = \frac{N}{|X|} = \frac{N}{r} = \beta$, hence Eq.(6) and Eq.(2) are also equivalent. It is important to emphasize, though, that this equivalence holds only for a specific choice of $\beta = n(x)$. While clearly the IB iterative algorithm (and problem) are meaningful for any value of $\beta$, there is no such freedom (for good or worse) in the ML setting, and the exponential factor in EM must be $n(x)$.

## 5 Comparing ML and IB

**Claim 5.1** *When $X$ is uniformly distributed and $r = |X|$, all the fixed points of the likelihood $L$ are mapped to all the fixed points of the IB-functional $\mathcal{L}$ with $\beta = n(x)$. Moreover,*

*at the fixed points, $-\log L \propto \mathcal{L} + k$, with $k$ constant.* [1]

**Corollary 5.2** *When $X$ is uniformly distributed, every algorithm which finds a fixed point of $L$, induces a fixed point of $\mathcal{L}$ with $\beta = n(x)$, and vice versa. When the algorithm finds several fixed points, the solution that maximizes $L$ is mapped to the one that minimizes $\mathcal{L}$.*

**Proof:** We prove the direction from ML to IB. the opposite direction is similar. We assume that we are given observations $n(x, y)$ where $n(x)$ is constant, and $\pi, \theta$ that define a fixed point of the likelihood $L$. As a result, this is also a fixed point of the EM algorithm (where $q_x(t)$ is defined through an $E$-step). Using observation 4.2 it follows that this fixed-point is mapped to a fixed-point of $\mathcal{L}$ with $\beta = n(x)$, as required.

Since at the fixed point, $-\log L = F$, it is enough to show the relationship between $F$ and $\mathcal{L}$. Rewriting $F$ from Eq.( 10) we get

$$F(n(x,y) : q, \pi, \theta) = \sum_{t,x} q_x(t) \log \frac{q_x(t)}{\pi(t)} - \sum_{t,y} \log \theta(y|t) \sum_x n(x,y) q_x(t) . \quad (13)$$

Using the $ML \rightarrow IB$ mapping and observation 4.1 we get

$$F = \sum_{t,x} q(t|x) \log \frac{q(t|x)}{q(t)} - r\beta \sum_{t,y} \log q(y|t) \sum_x p(x,y) q(t|x) . \quad (14)$$

Multiplying both sides by $p(x) = \frac{1}{|X|} = r^{-1}$ and using the IB Markovian independence relation, we find that

$$r^{-1}F \quad = \quad \sum_{t,x} p(x) q(t|x) \log \frac{q(t|x)}{q(t)} - \beta \sum_{t,y} q(t) q(y|t) \log q(y|t) . \quad (15)$$

Reducing a (constant) $\beta H(Y) = -\beta \sum_{t,y} q(t) q(y|t) \log p(y)$ to both sides gives:

$$r^{-1}F - \beta H(Y) = I(T;X) - \beta I(T;Y) = \mathcal{L} , \quad (16)$$

as required. We emphasize again that this equivalence is for a specific value of $\beta = n(x)$.

**Corollary 5.3** *When $X$ is uniformly distributed and $r = |X|$, every algorithm decreases $F$, iff it decreases $\mathcal{L}$ with $\beta = n(x)$.*

This corollary is a direct result from the above proof that showed the equivalence of the free energy of the model and the IB-functional (up to linear transformations).

The previous claims dealt with the special case of uniform prior over $X$. The following claims provide similar results for the general case, when the $N$ (or $\beta$) are large enough.

**Claim 5.4** *For $N \rightarrow \infty$ (or $\beta \rightarrow \infty$), all the fixed points of $L$ are mapped to all the fixed points of $\mathcal{L}$, and vice versa. Moreover, at the fixed points, $-\log L \propto \mathcal{L} + k$.*

**Corollary 5.5** *When $N \rightarrow \infty$ every algorithm which finds a fixed point of $L$, induces a fixed point of $\mathcal{L}$ with $\beta \rightarrow \infty$, and vice versa. When the algorithm finds several different fixed points, the solution that maximizes $L$ is mapped to the solution that minimize $\mathcal{L}$.*

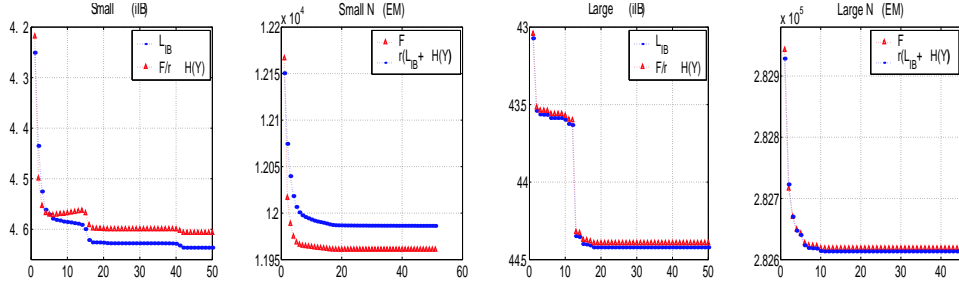

Figure 1: Progress of $\mathcal{L}$ and $F$ for different $\beta$ and $N$ values, while running iIB and EM.

**Proof:** Again, we prove only the direction from ML to IB as the opposite direction is similar. We are given $n(x, y)$ where $N = \sum_{x,y} n(x, y) \to \infty$ and $\pi, \theta$ that define a fixed point of $L$. Using the $E$-step in Eq.(6) we extract $q_x(t)$, ending up with a fixed point of the EM algorithm. We notice that from $N \to \infty$ follows $n(x) \to \infty \ \forall x \in X$. Therefore, the mapping $q_x(t)$ becomes deterministic:

$$q_x(t) = \begin{cases} 1 & t = argmin_{t'} D_{KL}(n(y|x)\|\theta(y|t')) \\ 0 & \text{otherwise.} \end{cases} \tag{17}$$

Performing the $ML \to IB$ mapping (including the IB-step), it is easy to verify that we get $q(y|t) = \theta(y|t)$ (but $q(t) \neq \pi(t)$ if the prior over $X$ is not uniform). After completing the mapping we try to update $q(t|x)$ through Eq.(2). Since now $\beta \to \infty$ it follows that $q(t|x)$ will remain deterministic. Specifically,

$$q^{new}(t|x) = \begin{cases} 1 & t = argmin_{t'} D_{KL}(p(y|x)\|q(y|t')) \\ 0 & \text{otherwise,} \end{cases} \tag{18}$$

which is equal to its previous value. Therefore, we are at a fixed point of the IB iterative algorithm, and by that at a fixed point of the IB-functional $\mathcal{L}$, as required.

To show that $-\log L \propto \mathcal{L} + k$ we notice again that at the fixed point $F = -\log L$. From Eq.(13) we see that

$$\lim_{N \to \infty} F = -\sum_{t,y} \log \theta(y|t) \sum_x n(x, y) q_x(t) \ . \tag{19}$$

Using the $ML \to IB$ mapping and similar algebra as above, we find that

$$\lim_{N \to \infty} F = \lim_{\beta \to \infty} -r\beta I(T; Y) + r\beta H(Y) = \lim_{\beta \to \infty} r(\mathcal{L} + \beta H(Y)) \ . \quad \square \tag{20}$$

**Corollary 5.6** *When $N \to \infty$ every algorithm decreases $F$ iff it decreases $\mathcal{L}$ with $\beta \to \infty$.*

How large must $N$ (or $\beta$) be? We address this question through numeric simulations. Yet, roughly speaking, we notice that the value of $N$ for which the above claims (approximately) hold is related to the "amount of uniformity" in $n(x)$. Specifically, a crucial step in the above proof assumed that each $n(x)$ is large enough such that $q_x(t)$ becomes deterministic. Clearly, when $n(x)$ is less uniform, achieving this situation requires larger $N$ values.

## 6   Simulations

We performed several different simulations using different IB and ML algorithms. Due to the lack of space, only one example is reported below; In this example we used the

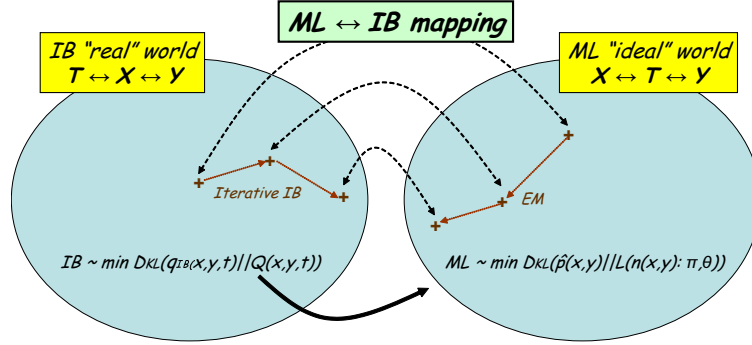

Figure 2: In general, ML (for mixture models) and IB operate in different solution spaces. Nonetheless, a sequence of probabilities that is obtained through some optimization routine (e.g., EM) in the "ML space", can be mapped to a sequence of probabilities in the "IB space", and vice versa. The main result of this paper is that under some conditions these two sequences are completely equivalent.

$Multi10_1$ subset of the 20-Newsgroups corpus [9], consisted of $500$ documents randomly chosen from 10 different discussion groups. Denoting the documents by $X$ and the words by $Y$, after pre-processing [10] we have $|X| = 500$, $|Y| = 2000$, $N = 43433$, $|T| = 10$. Since our main goal was to check the differences between IB and ML for different values of $N$ (or $\beta$), we further produced another dataset. In this data we randomly choose only about $5\%$ of the word occurrences for every document $x \in X$, ending up with $N = 2171$.

For both datasets we clustered the documents into 10 clusters, using both EM and the iterative IB (iIB) algorithm (where we took $p(x, y) = \frac{1}{N} n(x, y)$, $\beta = \frac{N}{r}$, $r = |X|$). For each algorithm we used the $ML \leftrightarrow IB$ mapping to calculate $F$ and $\mathcal{L}$ during the process (e.g., for iIB, after each iteration we mapped from $IB$ to $ML$, including the $M$-step, and calculated $F$). We repeated this procedure for 100 different initializations, for each dataset.

In these 200 runs we found that usually both algorithms improved both functionals monotonically. Comparing the functionals during the process, we see that for the smaller sample size the differences are indeed more evident (Figure 1). Comparing the final values of the functionals (after 50 iterations, which typically yielded convergence), we see that in $58$ out of 200 runs iIB converged to a smaller value of $F$ than EM. In 46 runs, EM converged to a smaller value of $\mathcal{L}$. Thus, occasionally, iIB finds a better ML solution or EM finds a better IB solution. This phenomenon was much more common for the large sample size case.

## 7   Discussion

While we have shown that the ML and IB approaches are equivalent under certain conditions, it is important to keep in mind the different assumptions both approaches make regarding the joint distribution over $x, y, t$. The mixture model (1) assumes that $Y$ is independent of $X$ given $T(X)$ and (2) assumes that $p(y|x)$ is one of a small number ($|T|$) of possible conditional distributions. For this reason, the marginal probability over $x, y$ (i.e., $p(x, y : \pi, \theta)$) is usually different from $\hat{p}(x, y) = \frac{1}{N} n(x, y)$. Indeed, an alternative view of ML estimation is as minimizing $D_{KL}(\hat{p}(x, y) \| L(n(x, y) : \pi, \theta))$.

On the other hand, in the IB framework, $T$ is *defined* through the IB Markovian independence relation: $T \leftarrow X \rightarrow Y$. Therefore, the solution space is the family of distributions for which this relation holds and the marginal distribution over $x, y$ is consistent with the input. Interestingly, it is possible to give an alternative formulation for the IB problem which

also involves KL minimization [11]. In this formulation the IB problem is related to minimizing $D_{KL}(q_{IB}(x,y,t)\|Q(x,y,t))$, where $Q(x,y,t)$ denotes the family of distributions for which the *mixture model* assumption holds, $X \rightarrow T \leftarrow Y$. [2]

In this sense, we may say that while solving the IB problem, one tries to minimize the KL with respect to the "ideal" world, in which $T$ separates $X$ from $Y$. On the other hand, while solving the ML problem, one assumes an "ideal" world, and tries to minimize the KL with respect to the given marginal distribution $\hat{p}(x,y)$. Our theoretical analysis shows that under the $ML \leftrightarrow IB$ mapping, these two procedures are in some cases equivalent (see Figure 2).

Once we are able to map between ML and IB, it should be interesting to try and adopt additional concepts from one approach to the other. In the following we provide two such examples. In the IB framework, for large enough $\beta$, the quality of a given solution is measured through $\frac{I(T;Y)}{I(X;Y)} \leq 1$ [1]. This measure provides a theoretical upper bound, which can be used for purposes of model selection and more. Using the $ML \leftrightarrow IB$ mapping, we can now adopt this measure for the ML estimation problem (for large enough $N$); In EM, the exponential factor $n(x)$ in general depends on $x$. However, its analogous component in the IB framework, $\beta$, obviously does not. Nonetheless, in principle it is possible to reformulate the IB problem while defining $\beta = \beta(x)$ (without changing the form of the optimal solution). We leave this issue for future research.

We have shown that for the multinomial mixture model, ML and IB are equivalent in some cases. It is worth noting that in principle, by choosing a different generative model, one may find further equivalences. Additionally, the IB method was recently extended into the multivariate case, where a new family of IB-like variational problems was presented and solved [11]. A natural question is to look for further generative models that can be mapped to this multivariate IB problems, and we are working in this direction.

## Acknowledgments

Insightful discussions with Nir Friedman, Naftali Tishby and Gal Elidan are greatly appreciated.

## Footnotes

[1] A similar result was recently obtained independently in [8] for the special case of 'hard" clustering. It is also important to keep in mind that in many clustering applications, a uniform prior over $X$ is 'forced" during the pre-process to avoid non-desirable bias. In particular this was done in several previous applications of the IB method (see [2] for details).

[2] The KL with respect to $Q$ is defined as the minimum over all the members in $Q$. Therefore, here, both arguments of the KL are changing during the process, and the distributions involved in the minimization are over all the three random variables.

## References

[1] N. Tishby, F. Pereira, and W. Bialek. The Information Bottleneck method. In *Proc. 37th Allerton Conference on Communication and Computation*, 1999.

[2] N. Slonim. The Information Bottleneck: theory and applications. Ph.D. thesis, The Hebrew University, 2002.

[3] T. M. Cover and J. A. Thomas. *Elements of Information Theory*. John Wiley & Sons, New York, 1991.

[4] T. Hofmann, J. Puzicha, and M. I. Jordan. Learning from dyadic data. In *Proc. of NIPS-11*, 1998.

[5] J. Puzicha, T. Hofmann, and J. M. Buhmann. Histogram clustering for unsupervised segmentation and image retrieval. In *Pattern Recognition Letters* 20(9), 899-909, 1999.

[6] A. P. Dempster, N. M. Laird, and D. B. Rubin. Maximum Likelihood from incomplete data via the EM algorithm. *Journal of the Royal Statistical Society B*, vol. 39, pp. 1-38, 1977.

[7] R. M. Neal and G. E. Hinton. A view of the EM algorithm that justifies incremental, sparse, and other variants. In M. I. Jordan (editor), *Learning in Graphical Models*, pp. 355-368, 1998.

[8] L. Hermes, T. zöller, and J. M. Buhmann. Parametric distributional clustering for image segmentation. In *Proc. of European Conference on Computer Vision (ECCV)*, 2002

[9] K. Lang. Learning to filter netnews. In *Proc. of the 12th Int. Conf. on Machine Learning*, 1995.

[10] N. Slonim, N. Friedman, and N. Tishby. Unsupervised document classification using sequential information maximization. In *Proc. of SIGIR-25*, 2002.

[11] N. Friedman, O. Mosenzon, N. Slonim, and N. Tishby. Multivariate Information Bottleneck. In *Proc. of UAI-17*, 2001.

